# From Isolation to Cooperation: An Alternative View of a System of Experts

**Stefan Schaal**[‡*]
sschaal@cc.gatech.edu
http://www.cc.gatech.edu/fac/Stefan.Schaal

**Christopher C. Atkeson**[‡]
cga@cc.gatech.edu
http://www.cc.gatech.edu/fac/Chris.Atkeson

[‡]College of Computing, Georgia Tech, 801 Atlantic Drive, Atlanta, GA 30332-0280
[*]ATR Human Information Processing, 2-2 Hikaridai, Seiko-cho, Soraku-gun, 619-02 Kyoto

## Abstract

We introduce a constructive, incremental learning system for regression problems that models data by means of locally linear experts. In contrast to other approaches, the experts are trained independently and do not compete for data during learning. Only when a prediction for a query is required do the experts cooperate by blending their individual predictions. Each expert is trained by minimizing a penalized local cross validation error using second order methods. In this way, an expert is able to find a local distance metric by adjusting the size and shape of the receptive field in which its predictions are valid, and also to detect relevant input features by adjusting its bias on the importance of individual input dimensions. We derive asymptotic results for our method. In a variety of simulations the properties of the algorithm are demonstrated with respect to interference, learning speed, prediction accuracy, feature detection, and task oriented incremental learning.

## 1. INTRODUCTION

Distributing a learning task among a set of experts has become a popular method in computational learning. One approach is to employ several experts, each with a *global* domain of expertise (e.g., Wolpert, 1990). When an output for a given input is to be predicted, every expert gives a prediction together with a confidence measure. The individual predictions are combined into a single result, for instance, based on a confidence weighted average. Another approach—the approach pursued in this paper—of employing experts is to create experts with *local* domains of expertise. In contrast to the global experts, the local experts have little overlap or no overlap at all. To assign a local domain of expertise to each expert, it is necessary to learn an expert selection system in addition to the experts themselves. This classifier determines which expert models are used in which part of the input space. For incremental learning, competitive learning methods are usually applied. Here the experts compete for data such that they change their domains of expertise until a stable configuration is achieved (e.g., Jacobs, Jordan, Nowlan, & Hinton, 1991). The advantage of local experts is that they can have simple parameterizations, such as locally constant or locally linear models. This offers benefits in terms of analyzability, learning speed, and robustness (e.g., Jordan & Jacobs, 1994). For simple experts, however, a large number of experts is necessary to model a function. As a result, the expert selection system has to be more complicated and, thus, has a higher risk of getting stuck in local minima and/or of learning rather slowly. In incremental learning, another potential danger arises when the input distribution of the data changes. The expert selection system usually makes either implicit or explicit prior assumptions about the input data distribution. For example, in the classical mixture model (McLachlan & Basford, 1988) which was employed in several local expert approaches, the prior probabilities of each mixture model can be interpreted as

the fraction of data points each expert expects to experience. Therefore, a change in input distribution will cause *all* experts to change their domains of expertise in order to fulfill these prior assumptions. This can lead to catastrophic interference.

In order to avoid these problems and to cope with the interference problems during incremental learning due to changes in input distribution, we suggest eliminating the competition among experts and instead isolating them during learning. Whenever some new data is experienced which is not accounted for by one of the current experts, a new expert is created. Since the experts do not compete for data with their peers, there is no reason for them to change the location of their domains of expertise. However, when it comes to making a prediction at a query point, all the experts cooperate by giving a prediction of the output together with a confidence measure. A blending of all the predictions of all experts results in the final prediction. It should be noted that these local experts combine properties of both the global and local experts mentioned previously. They act like global experts by learning independently of each other and by blending their predictions, but they act like local experts by confining themselves to a local domain of expertise, i.e., their confidence measures are large only in a local region.

The topic of data fitting with structurally simple local models (or experts) has received a great deal of attention in nonparametric statistics (e.g., Nadaraya, 1964; Cleveland, 1979; Scott, 1992, Hastie & Tibshirani, 1990). In this paper, we will demonstrate how a nonparametric approach can be applied to obtain the isolated expert network (Section 2.1), how its asymptotic properties can be analyzed (Section 2.2), and what characteristics such a learning system possesses in terms of the avoidance of interference, feature detection, dimensionality reduction, and incremental learning of motor control tasks (Section 3).

## 2. RECEPTIVE FIELD WEIGHTED REGRESSION

This paper focuses on regression problems, i.e., the learning of a map from $\Re^n \to \Re^m$. Each expert in our learning method, Receptive Field Weighted Regression (RFWR), consists of two elements, a locally linear model to represent the local functional relationship, and a receptive field which determines the region in input space in which the expert's knowledge is valid. As a result, a given data set will be modeled by piecewise linear elements, blended together. For 1000 noisy data points drawn from the unit interval of the function $z = \max[\exp(-10x^2), \exp(-50y^2), 1.25\exp(-5(x^2 + y^2))]$, Figure 1 illustrates an example of function fitting with RFWR. This function consists of a narrow and a wide ridge which are perpendicular to each other, and a Gaussian bump at the origin. Figure 1b shows the receptive fields which the system created during the learning process. Each experts' location is at the center of its receptive field, marked by a $\oplus$ in Figure 1b. The recep-

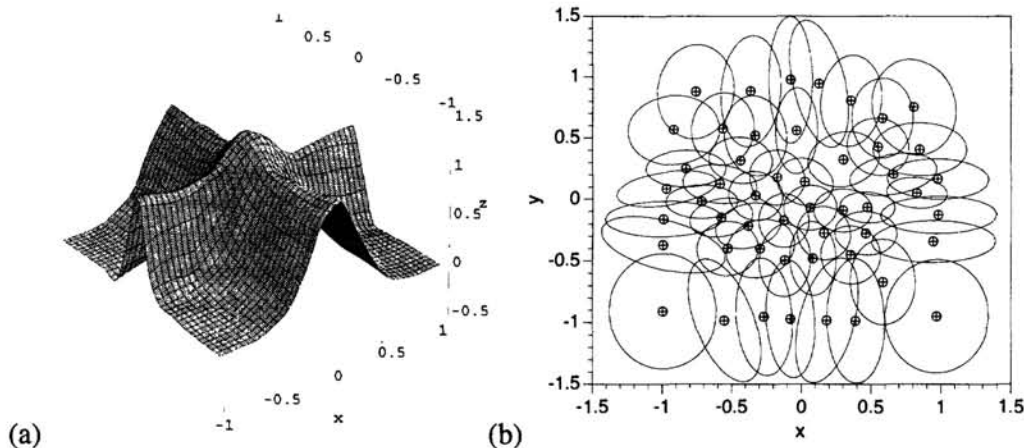

(a)                                                                    (b)

Figure 1: (a) result of function approximation with RFWR, (b) contour lines of 0.1 iso-activation of each expert in input space (the experts' centers are marked by small circles).

tive fields are modeled by Gaussian functions, and their 0.1 iso-activation lines are shown in Figure 1b as well. As can be seen, each expert focuses on a certain region of the input space, and the shape and orientation of this region reflects the function's complexity, or more precisely, the function's curvature, in this region. It should be noticed that there is a certain amount of overlap among the experts, and that the placement of experts occurred on a greedy basis during learning and is not globally optimal. The approximation result (Figure 1a) is a faithful reconstruction of the real function (MSE = 0.0025 on a test set, 30 epochs training, about 1 minute of computation on a SPARC10). As a baseline comparison, a similar result with a sigmoidal 3-layer neural network required about 100 hidden units and 10000 epochs of annealed standard backpropagation (about 4 hours on a SPARC10).

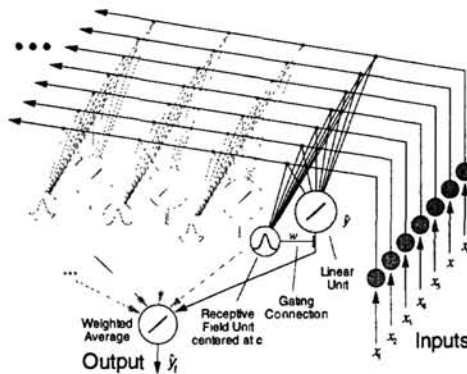

Figure 2: The RFWR network

## 2.1 THE ALGORITHM

RFWR can be sketched in network form as shown in Figure 2. All inputs connect to all expert networks, and new experts can be added as needed. Each expert is an independent entity. It consists of a two layer linear subnet and a receptive field subnet. The receptive field subnet has a single unit with a bell-shaped activation profile, centered at the fixed location $\mathbf{c}$ in input space. The maximal output of this unit is "1" at the center, and it decays to zero as a function of the distance from the center. For analytical convenience, we choose this unit to be Gaussian:

$$w = \exp\left(-\frac{1}{2}(\mathbf{x}-\mathbf{c})^T \mathbf{D}(\mathbf{x}-\mathbf{c})\right), \quad \text{where } \mathbf{D} = \mathbf{M}^T\mathbf{M} \tag{1}$$

$\mathbf{x}$ is the input vector, and $\mathbf{D}$ the distance metric, a positive definite matrix that is generated from the upper triangular matrix $\mathbf{M}$. The output of the linear subnet is:

$$\hat{y} = \mathbf{x}^T\mathbf{b} + b_0 = \tilde{\mathbf{x}}^T\beta \tag{2}$$

The connection strengths $\mathbf{b}$ of the linear subnet and its bias $b_0$ will be denoted by the $d$-dimensional vector $\beta$ from now on, and the tilde sign will indicate that a vector has been augmented by a constant "1", e.g., $\tilde{\mathbf{x}} = (\mathbf{x}^T, 1)^T$. In generating the total output, the receptive field units act as a gating component on the output, such that the total prediction is:

$$\hat{y}_t = \left(\sum_k w_k \hat{y}_k\right) \Big/ \left(\sum_k w_k\right) \tag{3}$$

The parameters $\beta$ and $\mathbf{M}$ are the primary quantities which have to be adjusted in the learning process: $\beta$ forms the locally linear model, while $\mathbf{M}$ determines the shape and orientation of the receptive fields. Learning is achieved by incrementally minimizing the cost function:

$$J = \left(\sum_i w_i (y_i - \hat{y}_{i,-i})^2\right) \Big/ \left(\sum_i w_i\right) + \gamma \sum_{n,m} D_{nm}^2 \tag{4}$$

The first term of this function is the weighted mean squared cross validation error over all experienced data points, a local cross validation measure (Schaal & Atkeson, 1994). The second term is a regularization or penalty term. Local cross validation by itself is consistent, i.e., with an increasing amount of data, the size of the receptive field of an expert would shrink to zero. This would require the creation of an ever increasing number of experts during the course of learning. The penalty term introduces some non-vanishing bias in each expert such that its receptive field size does not shrink to zero. By penalizing the squared coefficients of $\mathbf{D}$, we are essentially penalizing the second derivatives of the function at the site of the expert. This is similar to the approaches taken in spline fitting

(deBoor, 1978) and acts as a low-pass filter: the higher the second derivatives, the more smoothing (and thus bias) will be introduced. This will be analyzed further in Section 2.2.

The update equations for the linear subnet are the standard weighted recursive least squares equation with forgetting factor $\lambda$ (Ljung & Söderström, 1986):

$$\beta^{n+1} = \beta^n + w\,\mathbf{P}^{n+1}\tilde{\mathbf{x}}\,e_{cv}, \text{where } \mathbf{P}^{n+1} = \frac{1}{\lambda}\left(\mathbf{P}^n - \frac{\mathbf{P}^n\tilde{\mathbf{x}}\tilde{\mathbf{x}}^T\mathbf{P}^n}{\lambda/w + \tilde{\mathbf{x}}^T\mathbf{P}^n\tilde{\mathbf{x}}}\right) \text{ and } e_{cv} = \left(y - \tilde{\mathbf{x}}^T\beta^n\right) \tag{5}$$

This is a Newton method, and it requires maintaining the matrix $\mathbf{P}$, which is size $0.5d \times (d+1)$. The update of the receptive field subnet is a gradient descent in $J$:

$$\mathbf{M}^{n+1} = \mathbf{M}^n - \alpha\,\partial J/\partial\mathbf{M} \tag{6}$$

Due to space limitations, the derivation of the derivative in (6) will not be explained here. The major ingredient is to take this derivative as in a batch update, and then to reformulate the result as an iterative scheme. The derivatives in batch mode can be calculated exactly due to the Sherman-Morrison-Woodbury theorem (Belsley, Kuh, & Welsch, 1980; Atkeson, 1992). The derivative for the incremental update is a very good approximation to the batch update and realizes incremental local cross validation.

A new expert is initialized with a default $\mathbf{M}_{def}$ and all other variables set to zero, except the matrix $\mathbf{P}$. $\mathbf{P}$ is initialized as a diagonal matrix with elements $1/r_i^2$, where the $r_i$ are usually small quantities, e.g., 0.01. The $r_i$ are ridge regression parameters. From a probabilistic view, they are Bayesian priors that the $\beta$ vector is the zero vector. From an algorithmic view, they are fake data points of the form $[\mathbf{x} = (0,...,r_i^2,0,...)^T, y = 0]$ (Atkeson, Moore, & Schaal, submitted). Using the update rule (5), the influence of the ridge regression parameters would fade away due to the forgetting factor $\lambda$. However, it is useful to make the ridge regression parameters adjustable. As in (6), $r_i$ can be updated by gradient descent:

$$r_i^{n+1} = r_i^n - \alpha\,\partial J/\partial r_i \tag{7}$$

There are $d$ ridge regression parameters, one for each diagonal element of the $\mathbf{P}$ matrix. In order to add in the update of the ridge parameters as well as to compensate for the forgetting factor, an iterative procedure based on (5) can be devised which we omit here. The computational complexity of this update is much reduced in comparison to (5) since many computations involve multiplications by zero.

```
Initialize the RFWR network with no expert;
For every new training sample (x,y):
    a)    For k=1 to #experts:
              – calculate the activation from (1)
              – update the expert's parameters according to (5), (6), and (7)
          end;
    b)    If no expert was activated by more than w_gen:
              – create a new expert with c=x
          end;
    c)    If two experts are activated more than w_prune:
              – erase the expert with the smaller receptive field
          end;
    d)    calculate the mean, err_mean, and standard deviation err_std of the
          incrementally accumulated error err_k of all experts;
    e)    For k=1 to #experts:
              If (|err_k - err_mean| > θ err_std) reinitialize expert k with M = 2 * M_def
          end;
end;
```

In sum, a RFWR expert consists of three sets of parameters, one for the locally linear model, one for the size and shape of the receptive fields, and one for the bias. The linear model parameters are updated by a Newton method, while the other parameters are updated by gradient descent. In our implementations, we actually use second order gradient descent based on Sutton (1992), since, with minor extra effort, we can obtain estimates of the second derivatives of the cost function with respect to all parameters. Finally, the logic of RFWR becomes as shown in the pseudo-code above. Point c) and e) of the algorithm introduce a pruning facility. Pruning takes place either when two experts overlap too much, or when an expert has an exceptionally large mean squared error. The latter method corresponds to a simple form of outlier detection. Local optimization of a distance metric always has a minimum for a very large receptive field size. In our case, this would mean that an expert favors global instead of locally linear regression. Such an expert will accumulate a very large error which can easily be detected

in the given way. The mean squared error term, *err*, on which this outlier detection is based, is a bias-corrected mean squared error, as will be explained below.

## 2.2 ASYMPTOTIC BIAS AND PENALTY SELECTION

The penalty term in the cost function (4) introduces bias. In order to assess the asymptotic value of this bias, the real function $f(\mathbf{x})$, which is to be learned, is assumed to be represented as a Taylor series expansion at the center of an expert's receptive field. Without loss of generality, the center is assumed to be at the origin in input space. We furthermore assume that the size and shape of the receptive field are such that terms higher than $O(2)$ are negligible. Thus, the cost (4) can be written as:

$$J \approx \left( \int_{-\infty}^{+\infty} w \left( f_o + \mathbf{f}^T \mathbf{x} + \frac{1}{2} \mathbf{x}^T \mathbf{F} \mathbf{x} - b_o - \mathbf{b}^T \mathbf{x} \right)^2 dx \right) \Big/ \left( \int_{-\infty}^{+\infty} w \, dx \right) + \gamma \sum_{n,m} D_{nm} \tag{8}$$

where $f_0$, $\mathbf{f}$, and $\mathbf{F}$ denote the constant, linear, and quadratic terms of the Taylor series expansion, respectively. Inserting Equation (1), the integrals can be solved analytically after the input space is rotated by an orthonormal matrix transforming $\mathbf{F}$ to the diagonal matrix $\mathbf{F}'$. Subsequently, $b_0$, $\mathbf{b}$, and $\mathbf{D}$ can be determined such that $J$ is minimized:

$$b_0^* = f_0 + bias = f_0 + \frac{\gamma^{0.25}}{2^{0.75}} \sum_n \left( \text{sgn}(F_{nn}') \sqrt{|F_{nn}'|} \right), \quad \mathbf{b}^* = \mathbf{f}, \quad D_{nn}'^* = \frac{\sqrt{|F_{nn}'|}}{(2\gamma)^2} \tag{9}$$

This states that the linear model will asymptotically acquire the correct locally linear model, while the constant term will have bias proportional to the square root of the sum of the eigenvalues of $\mathbf{F}$, i.e., the $F_{nn}'$. The distance metric $\mathbf{D}$, whose diagonalized counterpart is $\mathbf{D}'$, will be a scaled image of the Hessian $\mathbf{F}$ with an additional square root distortion. Thus, the penalty term accomplishes the intended task: it introduces more smoothing the higher the curvature at an expert's location is, and it prevents the receptive field of an expert shrinking to zero size (which would obviously happen for $\gamma \to 0$). Additionally, Equation (9) shows how to determine $\gamma$ for a given learning problem from an estimate of the eigenvalues and a permissible bias. Finally, it is possible to derive estimates of the bias and the mean squared error of each expert from the current distance metric $\mathbf{D}$:

$$bias_{est} = \sqrt{0.5\gamma} \sum_n \left[ eigenvalues(D) \right]_n; \quad err_{est}^2 = \gamma \sum_{n,m} D_{nm}^2 \tag{10}$$

The latter term was incorporated in the mean squared error, *err*, in Section 2.1. Empirical evaluations (not shown here) verified the validity of these asymptotic results.

## 3. SIMULATION RESULTS

This section will demonstrate some of the properties of RFWR. In all simulations, the threshold parameters of the algorithm were set to $\theta = 3.5$, $w_{prune} = 0.9$, and $w_{min} = 0.1$. These quantities determine the overlap of the experts as well as the outlier removal threshold; the results below are not affected by moderate changes in these parameters.

### 3.1 AVOIDING INTERFERENCE

In order to test RFWR's sensitivity with respect to changes in input data distribution, the data of the example of Figure 1 was partitioned into three separate training sets $T_1 = \{(x, y, z) \mid -1.0 < x < -0.2\}$, $T_2 = \{(x, y, z) \mid -0.4 < x < 0.4\}$, $T_3 = \{(x, y, z) \mid 0.2 < x < 1.0\}$. These data sets correspond to three overlapping stripes of data, each having about 400 uniformly distributed samples. From scratch, a RFWR network was trained first on $T_1$ for 20 epochs, then on $T_2$ for 20 epochs, and finally on $T_3$ for 20 epochs. The penalty was chosen as in the example of Figure 1 to be $\gamma = 1.e - 7$, which corresponds to an asymptotic bias of

0.1 at the sharp ridge of the function. The default distance metric **D** was 50\***I**, where **I** is the identity matrix. Figure 3 shows the results of this experiment. Very little interference can be found. The MSE on the test set increased from 0.0025 (of the original experiment of Figure 1) to 0.003, which is still an excellent reconstruction of the real function.

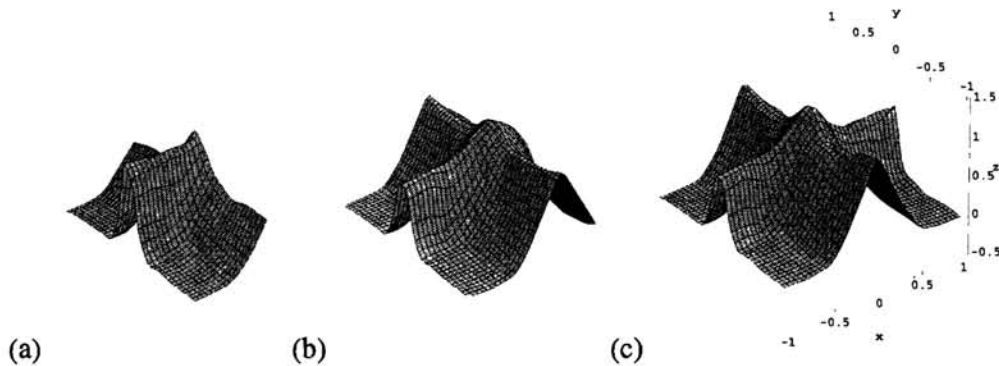

(a)                               (b)                               (c)

Figure 3: Reconstructed function after training on (a) $T_1$, (b) then $T_2$, (c) and finally $T_3$.

## 3.2 LOCAL FEATURE DETECTION

The examples of RFWR given so far did not require ridge regression parameters. Their importance, however, becomes obvious when dealing with locally rank deficient data or with irrelevant input dimensions. A learning system should be able to recognize irrelevant input dimensions. It is important to note that this cannot be accomplished by a distance metric. The distance metric is only able to decide to what spatial extent averaging over data in a certain dimension should be performed. However, the distance metric has no means to exclude an input dimension. In contrast, bias learning with ridge regression parameters is able to exclude input dimensions. To demonstrate this, we added 8 purely noisy inputs (N(0,0.3)) to the data drawn from the function of Figure 1. After 30 epochs of training on a 10000 data point training set, we analyzed histograms of the order of magnitude of the ridge regression parameters in all 10+bias input dimensions over all the 79 experts that had been generated by the learning algorithm. All experts recognized that the input dimensions 3 to 8 did not contain relevant information, and correctly increased the corresponding ridge parameters to large values. The effect of a large ridge regression parameter is that the associated regression coefficient becomes zero. In contrast, the ridge parameters of the inputs 1, 2, and the bias input remained very small. The MSE on the test set was 0.0026, basically identical to the experiment with the original training set.

## 3.3 LEARNING AN INVERSE DYNAMICS MODEL OF A ROBOT ARM

Robot learning is one of the domains where incremental learning plays an important role. A real movement system experiences data at a high rate, and it should incorporate this data immediately to improve its performance. As learning is task oriented, input distributions will also be task oriented and interference problems can easily arise. Additionally, a real movement system does not sample data from a training set but rather has to move in order to receive new data. Thus, training data is always temporally correlated, and learning must be able to cope with this. An example of such a learning task is given in Figure 4 where a simulated 2 DOF robot arm has to learn to draw the figure "8" in two different regions of the work space at a moderate speed (1.5 sec duration). In this example, we assume that the correct movement plan exists, but that the inverse dynamics model which is to be used to control this movement has not been acquired. The robot is first trained for 10 minutes (real movement time) in the region of the lower target trajectory where it performs a variety of rhythmic movements under simple PID control. The initial performance of this controller is shown in the bottom part of Figure 4a. This training enables the robot to learn the locally appropriate inverse dynamics model, a $\Re^6 \to \Re^2$ continuous mapping. Subsequent per-

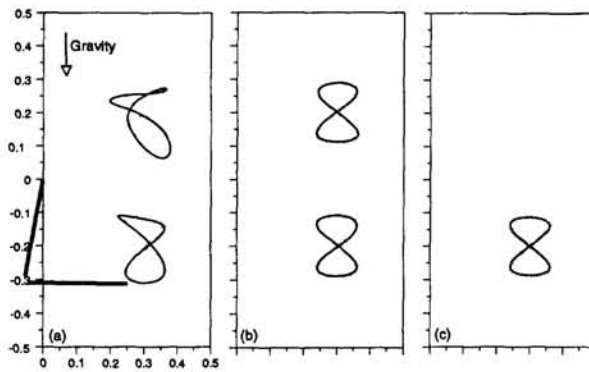

Figure 4: Learning to draw the figure "8" with a 2-joint arm: (a) Performance of a PID controller before learning (the dimmed lines denote the desired trajectories, the solid lines the actual performance); (b) Performance after learning using a PD controller with feedforward commands from the learned inverse model; (c) Performance of the learned controller after training on the upper "8" of (b) (see text for more explanations).

formance using this inverse model for control is depicted in the bottom part of Figure 4b. Afterwards, the same training takes place in the region of the upper target trajectory in order to acquire the inverse model in this part of the world. The figure "8" can then equally well be drawn there (upper part of Figure 4a,b). Switching back to the bottom part of the work space (Figure 4c), the first task can still be performed as before. No interference is recognizable. Thus, the robot could learn fast and reliably to fulfill the two tasks. It is important to note that the data generated by the training movements did not always have locally full rank. All the parameters of RFWR were necessary to acquire the local inverse model appropriately. A total of 39 locally linear experts were generated.

## 4. DISCUSSION

We have introduced an incremental learning algorithm, RFWR, which constructs a network of isolated experts for supervised learning of regression tasks. Each expert determines a locally linear model, a local distance metric, and local bias parameters by incrementally minimizing a penalized local cross validation error. Our algorithm differs from other local learning techniques by entirely avoiding competition among the experts, and by being based on nonparametric instead of parametric statistics. The resulting properties of RFWR are a) avoidance of interference in the case of changing input distributions, b) fast incremental learning by means of Newton and second order gradient descent methods, c) analyzable asymptotic properties which facilitate the selection of the fit parameters, and d) local feature detection and dimensionality reduction. The isolated experts are also ideally suited for parallel implementations. Future work will investigate computationally less costly delta-rule implementations of RFWR, and how well RFWR scales in higher dimensions.

## 5. REFERENCES

Atkeson, C. G., Moore, A. W., & Schaal, S. (submitted). "Locally weighted learning." *Artificial Intelligence Review*.

Atkeson, C. G. (1992). "Memory-based approaches to approximating continuous functions." In: Casdagli, M., & Eubank, S. (Eds.), *Nonlinear Modeling and Forecasting*, pp.503-521. Addison Wesley.

Belsley, D. A., Kuh, E., & Welsch, R. E. (1980). *Regression diagnostics: Identifying influential data and sources of collinearity*. New York: Wiley.

Cleveland, W. S. (1979). "Robust locally weighted regression and smoothing scatterplots." *J. American Stat. Association*, **74**, pp.829-836.

de Boor, C. (1978). *A practical guide to splines*. New York: Springer.

Hastie, T. J., & Tibshirani, R. J. (1990). *Generalized additive models*. London: Chapman and Hall.

Jacobs, R. A., Jordan, M. I., Nowlan, S. J., & Hinton, G. E. (1991). "Adaptive mixtures of local experts." *Neural Computation*, **3**, pp.79-87.

Jordan, M. I., & Jacobs, R. (1994). "Hierarchical mixtures of experts and the EM algorithm." *Neural Computation*, **6**, pp.79-87.

Ljung, L., & S_derstr_m, T. (1986). *Theory and practice of recursive identification*. Cambridge, MIT Press.

McLachlan, G. J., & Basford, K. E. (1988). *Mixture models*. New York: Marcel Dekker.

Nadaraya, E. A. (1964). "On estimating regression." *Theor. Prob. Appl.*, **9**, pp.141-142.

Schaal, S., & Atkeson, C. G. (1994b). "Assessing the quality of learned local models." In: Cowan, J. , Tesauro, G., & Alspector, J. (Eds.), *Advances in Neural Information Processing Systems 6*. Morgan Kaufmann.

Scott, D. W. (1992). *Multivariate Density Estimation*. New York: Wiley.

Sutton, R. S. (1992). "Gain adaptation beats least squares." In: *Proc. of 7th Yale Workshop on Adaptive and Learning Systems*, New Haven, CT.

Wolpert, D. H. (1990). "Stacked genealization." Los Alamos Technical Report LA-UR-90-3460 .
